# Compressed Least-Squares Regression

**Odalric-Ambrym Maillard** and **Rémi Munos**
SequeL Project, INRIA Lille - Nord Europe, France
{odalric.maillard, remi.munos}@inria.fr

## Abstract

We consider the problem of learning, from $K$ data, a regression function in a linear space of high dimension $N$ using projections onto a random subspace of lower dimension $M$. From any algorithm minimizing the (possibly penalized) empirical risk, we provide bounds on the excess risk of the estimate computed in the projected subspace (compressed domain) in terms of the excess risk of the estimate built in the high-dimensional space (initial domain). We show that solving the problem in the compressed domain instead of the initial domain reduces the estimation error at the price of an increased (but controlled) approximation error. We apply the analysis to Least-Squares (LS) regression and discuss the excess risk and numerical complexity of the resulting "Compressed Least Squares Regression" (CLSR) in terms of $N$, $K$, and $M$. When we choose $M = O(\sqrt{K})$, we show that CLSR has an estimation error of order $O(\log K/\sqrt{K})$.

## 1 Problem setting

We consider a regression problem where we observe data $\mathcal{D}_K = (\{x_k, y_k\}_{k \leq K})$ (where $x_k \in \mathcal{X}$ and $y_k \in \mathbb{R}$) are assumed to be independently and identically distributed (i.i.d.) from some distribution $P$, where $x_k \sim P_{\mathcal{X}}$ and $y_k = f^*(x_k) + \eta_k(x_k)$, where $f^*$ is the (unknown) target function, and $\eta_k$ a centered independent noise of variance $\sigma^2(x_k)$. For a given class of functions $\mathcal{F}$, and $f \in \mathcal{F}$, we define the empirical (quadratic) error

$$L_K(f) \stackrel{\text{def}}{=} \frac{1}{K} \sum_{k=1}^{K} [y_k - f(x_k)]^2,$$

and the generalization (quadratic) error

$$L(f) \stackrel{\text{def}}{=} \mathbb{E}_{(X,Y) \sim P}[(Y - f(X))^2].$$

Our goal is to return a regression function $\widehat{f} \in \mathcal{F}$ with lowest possible generalization error $L(\widehat{f})$.

**Notations:** *In the sequel we will make use of the following notations about norms: for $h : \mathcal{X} \mapsto \mathbb{R}$, we write $||h||_P$ for the $L_2$ norm of $h$ with respect to (w.r.t.) the measure $P$, $||h||_{P_K}$ for the $L_2$ norm of $h$ w.r.t. the empirical measure $P_K$, and for $u \in \mathbb{R}^n$, $||u||$ denotes by default $\left( \sum_{i=1}^{n} u_i^2 \right)^{1/2}$.*

The measurable function minimizing the generalization error is $f^*$, but it may be the case that $f^* \notin \mathcal{F}$. For any regression function $\widehat{f}$, we define the **excess risk**

$$L(\widehat{f}) - L(f^*) = ||\widehat{f} - f^*||_P^2,$$

which decomposes as the sum of the **estimation error** $L(\widehat{f}) - \inf_{f \in \mathcal{F}} L(f)$ and the **approximation error** $\inf_{f \in \mathcal{F}} L(f) - L(f^*) = \inf_{f \in \mathcal{F}} ||f - f^*||_P^2$ which measures the distance between $f^*$ and the function space $\mathcal{F}$.

In this paper we consider a class of linear functions $\mathcal{F}_N$ defined as the span of a set of $N$ functions $\{\varphi_n\}_{1 \leq n \leq N}$ called *features*. Thus: $\mathcal{F}_N \overset{\text{def}}{=} \{f_\alpha \overset{\text{def}}{=} \sum_{n=1}^N \alpha_n \varphi_n, \; \alpha \in \mathbb{R}^N\}$.

When the number of data $K$ is larger than the number of features $N$, the ordinary Least-Squares Regression (LSR) provides the LS solution $f_{\widehat{\alpha}}$ which is the minimizer of the empirical risk $L_K(f)$ in $\mathcal{F}_N$. Note that here $L_K(f_\alpha)$ rewrites $\frac{1}{K}||\Phi\alpha - Y||_K$ where $\Phi$ is the $K \times N$ matrix with elements $(\varphi_n(x_k))_{1 \leq n \leq N, 1 \leq k \leq K}$ and $Y$ the $K$-vector with components $(y_k)_{1 \leq k \leq K}$.

Usual results provide bound on the estimation error as a function of the capacity of the function space and the number of data. In the case of linear approximation, the capacity measures (such as covering numbers [23] or the pseudo-dimension [16]) depend on the number of features (for example the pseudo-dimension is at most $N + 1$). For example, let $f_{\widehat{\alpha}}$ be a LS estimate (minimizer of $L_K$ in $\mathcal{F}_N$), then (a more precise statement will be stated later in Subsection 3) the expected estimation error is bounded as:

$$\mathbb{E}\big[L(f_{\widehat{\alpha}}) - \inf_{f \in \mathcal{F}_N} L(f)\big] \leq c\sigma^2 \frac{N \log K}{K}, \tag{1}$$

where $c$ is a universal constant, $\sigma \overset{\text{def}}{=} \sup_{x \in \mathcal{X}} \sigma(x)$, and the expectation is taken with respect to $P$. Now, the excess risk is the sum of this estimation error and the approximation error $\inf_{f \in \mathcal{F}_N} ||f - f^*||_P$ of the class $\mathcal{F}_N$. Since the later usually decreases when the number of features $N$ increases [13] (e.g. when $\bigcup_N \mathcal{F}_N$ is dense in $L_2(P)$), we see the usual tradeoff between small estimation error (low $N$) and small approximation error (large $N$).

In this paper we are interested in the setting when $N$ is large so that the approximation error is small. Whenever $N$ is larger than $K$ we face the overfitting problem since there are more parameters than actual data (more variables than constraints), which is illustrated in the bound (1) which provides no information about the generalization ability of any LS estimate. In addition, there are many minimizers (in fact a vector space of same dimension as the null space of $\Phi^T\Phi$) of the empirical risk. To overcome the problem, several approaches have been proposed in the literature:

- **LS solution with minimal norm**: The solution is the minimizer of the empirical error with minimal ($l_1$ or $l_2$)-norm: $\widehat{\alpha} = \arg\min_{\Phi\alpha=Y} ||\alpha||_{1 \text{ or } 2}$, (or a robust solution $\arg\min_{||\Phi\alpha-Y||_2 \leq \varepsilon} ||\alpha||_1$). The choice of $\ell_2$-norm yields the ordinary LS solution. The choice of $\ell_1$-norm has been used for generating sparse solutions (e.g. the Basis Pursuit [10]), and assuming that the target function admits a sparse decomposition, the field of Compressed Sensing [9, 21] provides sufficient conditions for recovering the exact solution. However, such conditions (e.g. that $\Phi$ possesses a Restricted Isometric Property (RIP)) does not hold in general in this regression setting. On another aspect, solving these problems (both for $l_1$ or $l_2$-norm) when $N$ is large is numerically expensive.

- **Regularization**. The solution is the minimizer of the empirical error plus a penalty term, for example

$$\widehat{f} = \arg\min_{f \in \mathcal{F}_N} L_K(f) + \lambda ||f||_p^p, \quad \text{for } p = 1 \text{ or } 2.$$

  where $\lambda$ is a parameter and usual choices for the norm are $\ell_2$ (ridge-regression [20]) and $\ell_1$ (LASSO [19]). A close alternative is the Dantzig selector [8, 5] which solves: $\widehat{\alpha} = \arg\min_{||\alpha||_1 \leq \lambda} ||\Phi^T(Y - \Phi\alpha)||_\infty$. The numerical complexity and generalization bounds of those methods depend on the sparsity of the target function decomposition in $\mathcal{F}_N$.

Now if we possess a sequence of function classes $(\mathcal{F}_N)_{N \geq 1}$ with increasing capacity, we may perform **structural risk minimization** [22] by solving in each model the empirical risk penalized by a term that depends on the size of the model: $\widehat{f}_N = \arg\min_{f \in \mathcal{F}_N, N \geq 1} L_K(f) + \text{pen}(N, K)$, where the penalty term measures the capacity of the function space.

In this paper we follow another approach where instead of searching in the large space $\mathcal{F}_N$ (where $N > K$) for a solution that minimizes the empirical error plus a penalty term, we simply search for the empirical error minimizer in a (randomly generated) lower dimensional subspace $\mathcal{G}_M \subset \mathcal{F}_N$ (where $M < K$).

**Our contribution:** We consider a set of $M$ random linear combinations of the initial $N$ features and perform our favorite LS regression algorithm (possibly regularized) using those "compressed

features". This is equivalent to projecting the $K$ points $\{\varphi(x_k) \in \mathbb{R}^N, k = 1..K\}$ from the initial domain (of size $N$) onto a random subspace of dimension $M$, and then performing the regression in the "compressed domain" (i.e. span of the compressed features). This is made possible because random projections approximately preserve inner products between vectors (by a variant of the Johnson-Lindenstrauss Lemma stated in Proposition 1.

Our main result is a bound on the excess risk of a linear estimator built in the compressed domain in terms of the excess risk of the linear estimator built in the initial domain (Section 2). We further detail the case of ordinary Least-Squares Regression (Section 3) and discuss, in terms of $M$, $N$, $K$, the different tradeoffs concerning the *excess risk* (reduced estimation error in the compressed domain versus increased approximation error introduced by the random projection) and the *numerical complexity* (reduced complexity of solving the LSR in the compressed domain versus the additional load of performing the projection).

As a consequence, we show that by choosing $M = O(\sqrt{K})$ projections we define a **Compressed Least-Squares Regression** which uses $O(NK^{3/2})$ elementary operations to compute a regression function with estimation error (relatively to the initial function space $\mathcal{F}_N$) of order $\log K / \sqrt{K}$ up to a multiplicative factor which depends on the best approximation of $f^*$ in $\mathcal{F}_N$. This is competitive with the best methods, up to our knowledge.

**Related works:** Using dimension reduction and random projections in various learning areas has received considerable interest over the past few years. In [7], the authors use a SVM algorithm in a compressed space for the purpose of classification and show that their resulting algorithm has good generalization properties. In [25], the authors consider a notion of compressed linear regression. For data $Y = X\beta + \varepsilon$, where $\beta$ is the target and $\varepsilon$ a standard noise, they use compression of the set of data, thus considering $AY = AX\beta + A\varepsilon$, where $A$ has a Restricted Isometric Property. They provide an analysis of the LASSO estimator built from these compressed data, and discuss a property called sparsistency, i.e. the number of random projections needed to recover $\beta$ (with high probability) when it is sparse. These works differ from our approach in the fact that we do not consider a compressed (input and/or output) data space but a compressed feature space instead.

In [11], the authors discuss how compressed measurements may be useful to solve many detection, classification and estimation problems without having to reconstruct the signal ever. Interestingly, they make no assumption about the signal being sparse, like in our work. In [6, 17], the authors show how to map a kernel $k(x, y) = \varphi(x) \cdot \varphi(y)$ into a low-dimensional space, while still approximately preserving the inner products. Thus they build a low-dimensional feature space specific for (translation invariant) kernels.

## 2 Linear regression in the compressed domain

We remind that the initial set of features is $\{\varphi_n : \mathcal{X} \mapsto \mathbb{R}, 1 \leq n \leq N\}$ and the initial domain $\mathcal{F}_N \stackrel{\text{def}}{=} \{f_\alpha = \sum_{n=1}^N \alpha_n \varphi_n, \alpha \in \mathbb{R}^N\}$ is the span of those features. We write $\varphi(x)$ the $N$-vector of components $(\varphi_n(x))_{n \leq N}$. Let us now define the random projection. Let $A$ be a $M \times N$ matrix of i.i.d. elements drawn for some distribution $\rho$. Examples of distributions are:

- Gaussian random variables $\mathcal{N}(0, 1/M)$,
- $\pm$ Bernoulli distributions, i.e. which takes values $\pm 1/\sqrt{M}$ with equal probability $1/2$,
- Distribution taking values $\pm\sqrt{3/M}$ with probability $1/6$ and $0$ with probability $2/3$.

The following result (proof in the supplementary material) states the property that inner-product are approximately preserved through random projections (this is a simple consequence of the Johnson-Lindenstrauss Lemma):

**Proposition 1** *Let $(u_k)_{1 \leq k \leq K}$ and $v$ be vectors of $\mathbb{R}^N$. Let $A$ be a $M \times N$ matrix of i.i.d. elements drawn from one of the previously defined distributions. For any $\varepsilon > 0$, $\delta > 0$, for $M \geq \frac{1}{\frac{\varepsilon^2}{4} - \frac{\varepsilon^3}{6}} \log \frac{4K}{\delta}$, we have, with probability at least $1 - \delta$, for all $k \leq K$,*

$$|Au_k \cdot Av - u_k \cdot v| \leq \varepsilon ||u_k|| \, ||v||.$$

We now introduce the set of $M$ **compressed features** $(\psi_m)_{1 \leq m \leq M}$ such that $\psi_m(x) \overset{\text{def}}{=} \sum_{n=1}^{N} A_{m,n} \varphi_n(x)$. We also write $\psi(x)$ the $M$-vector of components $(\psi_m(x))_{m \leq M}$. Thus $\psi(x) = A\varphi(x)$. We define the **compressed domain** $\mathcal{G}_M \overset{\text{def}}{=} \{g_\beta = \sum_{m=1}^{M} \beta_m \psi_m, \ \beta \in \mathbb{R}^M\}$ the span of the compressed features (vector space of dimension at most $M$). Note that each $\psi_m \in \mathcal{F}_N$, thus $\mathcal{G}_M$ is a subspace of $\mathcal{F}_N$.

## 2.1 Approximation error

We now compare the approximation error assessed in the compressed domain $\mathcal{G}_M$ versus in the initial space $\mathcal{F}_N$. This applies to the linear algorithms mentioned in the introduction such as ordinary LS regression (analyzed in details in Section 3), but also its penalized versions, e.g. LASSO and ridge regression. Define $\alpha^+ = \arg\min_{\alpha \in \mathbb{R}^N} L(f_\alpha) - L(f^*)$ the parameter of the best regression function in $\mathcal{F}_N$.

**Theorem 1** *For any $\delta > 0$, any $M \geq 15 \log(8K/\delta)$, let $A$ be a random $M \times N$ matrix defined like in Proposition 1, and $\mathcal{G}_M$ be the compressed domain resulting from this choice of $A$. Then with probability at least $1 - \delta$,*

$$\inf_{g \in \mathcal{G}_M} ||g - f^*||_P^2 \leq \frac{8 \log(8K/\delta)}{M} ||\alpha^+||^2 \left( \mathbb{E}[||\varphi(X)||^2] + 2 \sup_{x \in \mathcal{X}} ||\varphi(x)||^2 \sqrt{\frac{\log 4/\delta}{2K}} \right) + \inf_{f \in \mathcal{F}_N} ||f - f^*||_P^2. \tag{2}$$

This theorem shows the tradeoff in terms of estimation and approximation errors for an estimator $\widehat{g}$ obtained in the compressed domain compared to an estimator $\widehat{f}$ obtained in the initial domain:

- Bounds on the estimation error of $\widehat{g}$ in $\mathcal{G}_M$ are usually smaller than that of $\widehat{f}$ in $\mathcal{F}_N$ when $M < N$ (since the capacity of $\mathcal{F}_N$ is larger than that of $\mathcal{G}_M$).
- Theorem 1 says that the approximation error assessed in $\mathcal{G}_M$ increases by at most $O(\frac{\log(K/\delta)}{M})||\alpha^+||^2 \mathbb{E}||\varphi(X)||^2$ compared to that in $\mathcal{F}_N$.

*Proof:* Let us write $f^+ \overset{\text{def}}{=} f_{\alpha^+} = \arg\min_{f \in \mathcal{F}_N} ||f - f^*||_P$ and $g^+ \overset{\text{def}}{=} g_{A\alpha^+}$. The approximation error assessed in the compressed domain $\mathcal{G}_M$ is bounded as

$$\inf_{g \in \mathcal{G}_M} ||g - f^*||_P^2 \quad \leq \quad ||g^+ - f^*||_P^2 = ||g^+ - f^+||_P^2 + ||f^+ - f^*||_P^2, \tag{3}$$

since $f^+$ is the orthogonal projection of $f^*$ on $\mathcal{F}_N$ and $g^+$ belongs to $\mathcal{F}_N$. We now bound $||g^+ - f^+||_P^2$ using concentration inequalities. Define $Z(x) \overset{\text{def}}{=} A\alpha^+ \cdot A\varphi(x) - \alpha^+ \cdot \varphi(x)$. Define $\varepsilon^2 \overset{\text{def}}{=} \frac{8}{M} \log(8K/\delta)$. For $M \geq 15 \log(8K/\delta)$ we have $\varepsilon < 3/4$ thus $M \geq \frac{\log(8K/\delta)}{\varepsilon^2/4 - \varepsilon^3/6}$. Proposition 1 applies and says that on an event $\mathcal{E}$ of probability at least $1 - \delta/2$, we have for all $k \leq K$,

$$|Z(x_k)| \leq \varepsilon ||\alpha^+|| \, ||\varphi(x_k)|| \leq \varepsilon ||\alpha^+|| \sup_{x \in \mathcal{X}} ||\varphi(x)|| \overset{\text{def}}{=} C \tag{4}$$

On the event $\mathcal{E}$, we have with probability at least $1 - \delta'$,

$$
\begin{aligned}
||g^+ - f^+||_P^2 &= \mathbb{E}_{X \sim P_\mathcal{X}} |Z(X)|^2 \leq \frac{1}{K} \sum_{k=1}^{K} |Z(x_k)|^2 + C^2 \sqrt{\frac{\log(2/\delta')}{2K}} \\
&\leq \varepsilon^2 ||\alpha^+||^2 \left( \frac{1}{K} \sum_{k=1}^{K} ||\varphi(x_k)||^2 + \sup_{x \in \mathcal{X}} ||\varphi(x)||^2 \sqrt{\frac{\log(2/\delta')}{2K}} \right) \\
&\leq \varepsilon^2 ||\alpha^+||^2 \left( \mathbb{E}[||\varphi(X)||^2] + 2 \sup_{x \in \mathcal{X}} ||\varphi(x)||^2 \sqrt{\frac{\log(2/\delta')}{2K}} \right).
\end{aligned}
$$

where we applied two times Chernoff-Hoeffding's inequality. Combining with (3), unconditioning, and setting $\delta' = \delta/2$ then with probability at least $(1 - \delta/2)(1 - \delta') \geq 1 - \delta$ we have (2). $\qquad \square$

## 2.2 Computational issues

We now discuss the relative computational costs of a given algorithm applied either in the initial or in the compressed domain. Let us write $\mathrm{Cx}(\mathcal{D}_K, \mathcal{F}_N, P)$ the complexity (e.g. number of elementary operations) of an algorithm $\mathcal{A}$ to compute the regression function $\widehat{f}$ when provided with the data $\mathcal{D}_K$ and function space $\mathcal{F}_N$.

We plot in the table below, both for the initial and the compressed versions of the algorithm $\mathcal{A}$, the order of complexity for (i) the cost for building the feature matrix, (ii) the cost for computing the estimator, (iii) the cost for making one prediction (i.e. computing $\widehat{f}(x)$ for any $x$):

|  | Initial domain | Compressed domain |
|---|---|---|
| Construction of the feature matrix | $NK$ | $NKM$ |
| Computing the regression function | $\mathrm{Cx}(\mathcal{D}_K, \mathcal{F}_N, P)$ | $\mathrm{Cx}(\mathcal{D}_K, \mathcal{G}_M, P)$ |
| Making one prediction | $N$ | $NM$ |

Note that the values mentioned for the compressed domain are upper-bounds on the real complexity and do not take into account the possible sparsity of the projection matrix $A$ (which would speed up matrix computations, see e.g. [2, 1]).

## 3 Compressed Least-Squares Regression

We now analyze the specific case of Least-Squares Regression.

### 3.1 Excess risk of ordinary Least Squares regression

In order to bound the estimation error, we follow the approach of [13] which truncates (up to the level $\pm L$ where $L$ is a bound, assumed to be known, on $||f^*||_\infty$) the prediction of the LS regression function. The ordinary LS regression provides the regression function $f_{\widehat{\alpha}}$ where

$$\widehat{\alpha} = \underset{\alpha \in \mathrm{argmin}_{\alpha' \in \mathbb{R}^N} ||Y - \Phi \alpha'||}{\mathrm{argmin}} ||\alpha||.$$

Note that $\Phi \Phi^T \widehat{\alpha} = \Phi^T Y$, hence $\widehat{\alpha} = \Phi^\dagger Y \in \mathbb{R}^N$ where $\Phi^\dagger$ is the Penrose pseudo-inverse of $\Phi$[1]. Then the truncated predictor is: $\widehat{f}_L(x) \stackrel{\text{def}}{=} T_L[f_{\widehat{\alpha}}(x)]$, where

$$T_L(u) \stackrel{\text{def}}{=} \begin{cases} u & \text{if } |u| \leq L, \\ L \, \mathrm{sign}(u) & \text{otherwise.} \end{cases}$$

Truncation after the computation of the parameter $\widehat{\alpha} \in \mathbb{R}^N$, which is the solution of an unconstrained optimization problem, is easier than solving an optimization problem under the constraint that $||\alpha||$ is small (which is the approach followed in [23]) and allows for consistency results and prediction bounds. Indeed, the excess risk of $\widehat{f}_L$ is bounded as

$$\mathbb{E}(||\widehat{f} - f^*||_P^2) \leq c' \max\{\sigma^2, L^2\} \frac{1 + \log K}{K} N + 8 \inf_{f \in \mathcal{F}_N} ||f - f^*||_P^2 \tag{5}$$

where a bound on $c'$ is 9216 (see [13]). We have a simpler bound when we consider the expectation $\mathbb{E}_Y$ conditionally on the input data:

$$\mathbb{E}_Y(||\widehat{f} - f^*||_{P_K}^2) \leq \sigma^2 \frac{N}{K} + \inf_{f \in \mathcal{F}} ||f - f^*||_{P_K}^2 \tag{6}$$

**Remark:** Note that because we use the quadratic loss function, by following the analysis in [3], or by deriving tight bounds on the Rademacher complexity [14] and following Theorem 5.2 of Koltchinskii's Saint Flour course, it is actually possible to state assumptions under which we can remove the $\log K$ term in (5). We will not further detail such bounds since our motivation here is not to provide the tightest possible bounds, but rather to show how the excess risk bound for LS regression in the initial domain extends to the compressed domain.

### 3.2 Compressed Least-Squares Regression (CLSR)

CLSR is defined as the ordinary LSR in the compressed domain. Let $\widehat{\beta} = \Psi^\dagger Y \in \mathbb{R}^M$, where $\Psi$ is the $K \times M$ matrix with elements $(\psi_m(x_k))_{1 \leq m \leq M, 1 \leq k \leq K}$. The CLSR estimate is defined as $\widehat{g}_L(x) \overset{\text{def}}{=} T_L[g_{\widehat{\beta}}(x)]$. From Theorem 1, (5) and (6), we deduce the following excess risk bounds for the CLSR estimate:

**Corollary 1** *For any $\delta > 0$, set $M = 8\frac{||\alpha^+||\sqrt{\mathbb{E}||\varphi(X)||^2}}{\max(\sigma, L)}\sqrt{\frac{K\log(8K/\delta)}{c'(1+\log K)}}$. Then whenever $M \geq 15\log(8K/\delta)$, with probability at least $1 - \delta$, the expected excess risk of the CLSR estimate is bounded as*

$$
\begin{aligned}
\mathbb{E}(||\widehat{g}_L - f^*||_P^2) \quad \leq \quad & 16\sqrt{c'}\max\{\sigma, L\}||\alpha^+||\sqrt{\mathbb{E}||\varphi(X)||^2}\sqrt{\frac{(1+\log K)\log(8K/\delta)}{K}} \\
& \times \Big(1 + \frac{\sup_x ||\varphi(x)||^2}{\mathbb{E}||\varphi(X)||^2}\sqrt{\frac{\log 4/\delta}{2K}}\Big) + 8 \inf_{f \in \mathcal{F}_N} ||f - f^*||_P^2.
\end{aligned}
\tag{7}
$$

*Now set $M = \frac{||\alpha^+||\sqrt{\mathbb{E}||\varphi(X)||^2}}{\sigma}\sqrt{8K\log(8K/\delta)}$. Assume $N > K$ and that the features $(\varphi_k)_{1 \leq k \leq K}$ are linearly independent. Then whenever $M \geq 15\log(8K/\delta)$, with probability at least $1 - \delta$, the expected excess risk of the CLSR estimate conditionally on the input samples is upper bounded as*

$$
\mathbb{E}_Y(||\widehat{g}_L - f^*||_{P_K}^2) \leq 4\sigma||\alpha^+||\sqrt{\mathbb{E}||\varphi(X)||^2}\sqrt{\frac{2\log(8K/\delta)}{K}}\Big(1 + \frac{\sup_x ||\varphi(x)||^2}{\mathbb{E}||\varphi(X)||^2}\sqrt{\frac{\log 4/\delta}{2K}}\Big).
$$

*Proof:* Whenever $M \geq 15\log(8K/\delta)$ we deduce from Theorem 1 and (5) that the excess risk of $\widehat{g}_L$ is bounded as

$$
\begin{aligned}
\mathbb{E}(||\widehat{g}_L - f^*||_P^2) \leq\, & c'\max\{\sigma^2, L^2\}\frac{1+\log K}{K}M \\
& + 8\Big[\frac{8\log(8K/\delta)}{M}||\alpha^+||^2\Big(\mathbb{E}||\varphi(X)||^2 + 2\sup_x ||\varphi(x)||^2\sqrt{\frac{\log 4/\delta}{2K}}\Big) + \inf_{f \in \mathcal{F}_N} ||f - f^*||_P^2\Big].
\end{aligned}
$$

By optimizing on $M$, we deduce (7). Similarly, using (6) we deduce the following bound on $\mathbb{E}_Y(||\widehat{g}_L - f^*||_{P_K}^2)$:

$$
\sigma^2\frac{M}{K} + \frac{8}{M}\log(8K/\delta)||\alpha^+||^2\Big(\mathbb{E}||\varphi(X)||^2 + 2\sup_x ||\varphi(x)||^2\sqrt{\frac{\log 4/\delta}{2K}}\Big) + \inf_{f \in \mathcal{F}_N} ||f - f^*||_{P_K}^2.
$$

By optimizing on $M$ and noticing that $\inf_{f \in \mathcal{F}_N} ||f - f^*||_{P_K}^2 = 0$ whenever $N > K$ and the features $(\varphi_k)_{1 \leq k \leq K}$ are linearly independent, we deduce the second result. $\square$

**Remark 1** *Note that the second term in the parenthesis of (7) is negligible whenever $K \gg \log 1/\delta$. Thus we have the expected excess risk*

$$
\mathbb{E}(||\widehat{g}_L - f^*||_P^2) = O\Big(||\alpha^+||\sqrt{\mathbb{E}||\varphi(X)||^2}\frac{\log K/\delta}{\sqrt{K}} + \inf_{f \in \mathcal{F}_N} ||f - f^*||_P^2\Big).
\tag{8}
$$

*The choice of $M$ in the previous corollary depends on $||\alpha^+||$ and $\mathbb{E}||\varphi(X)||$ which are a priori unknown (since $f^*$ and $P_\mathcal{X}$ are unknown). If we set $M$ independently of $||\alpha^+||$, then an additional multiplicative factor of $||\alpha^+||$ appears in the bound, and if we replace $\mathbb{E}||\varphi(X)||$ by its bound $\sup_x ||\varphi(x)||$ (which is known) then this latter factor will appear instead of the former in the bound.*

**Complexity of CLSR:** The complexity of LSR for computing the regression function in the compressed domain only depends on $M$ and $K$, and is (see e.g. [4]) $\text{Cx}(\mathcal{D}_K, \mathcal{G}_M, P) = O(MK^2)$ which is of order $O(K^{5/2})$ when we choose the optimized number of projections $M = O(\sqrt{K})$. However the leading term when using CLSR is the cost for building the $\Psi$ matrix: $O(NK^{3/2})$.

# 4 Discussion

## 4.1 The factor $||\alpha^+||\sqrt{\mathbb{E}||\varphi(X)||^2}$

In light of Corollary 1, the important factor which will determine whether the CLSR provides low generalization error or not is $||\alpha^+||\sqrt{\mathbb{E}||\varphi(X)||^2}$. This factor indicates that a good set of features (for CLSR) should be such that the norm of those features as well as the norm of the parameter $\alpha^+$ of the projection of $f^*$ onto the span of those features should be small. A natural question is whether this product can be made small for appropriate choices of features. We now provide two specific cases for which this is actually the case: (1) when the features are rescaled orthonormal basis functions, and (2) when the features are specific wavelet functions. In both cases, we relate the bound to an assumption of regularity on the function $f^*$, and show that the dependency w.r.t. $N$ decreases when the regularity increases, and may even vanish.

**Rescaled Orthonormal Features:** Consider a set of orthonormal functions $(\eta_i)_{i\geq 1}$ w.r.t a measure $\mu$, i.e. $\langle \eta_i, \eta_j \rangle_\mu = \delta_{i,j}$. In addition we assume that the law of the input data is dominated by $\mu$, i.e. $P_{\mathcal{X}} \leq C\mu$ where $C$ is a constant. For instance, this is the case when the set $\mathcal{X}$ is compact, $\mu$ is the uniform measure and $P_{\mathcal{X}}$ has bounded density.

We define the set of $N$ features as: $\varphi_i \stackrel{\text{def}}{=} c_i\eta_i$, where $c_i > 0$, for $i \in \{1,\ldots,N\}$. Then any $f \in \mathcal{F}_N$ decomposes as $f = \sum_{i=1}^{N} \langle f, \eta_i \rangle \eta_i = \sum_{i=1}^{N} \frac{b_i}{c_i}\varphi_i$, where $b_i \stackrel{\text{def}}{=} \langle f, \eta_i \rangle$. Thus we have: $||\alpha||^2 = \sum_{i=1}^{N}(\frac{b_i}{c_i})^2$ and $\mathbb{E}||\varphi||^2 = \sum_{i=1}^{N} c_i^2 \int_{\mathcal{X}} \eta_i^2(x)dP_{\mathcal{X}}(x) \leq C\sum_{i=1}^{N} c_i^2$. Thus $||\alpha^+||^2\mathbb{E}||\varphi||^2 \leq C\sum_{i=1}^{N}(\frac{b_i}{c_i})^2 \sum_{i=1}^{N} c_i^2$.

Now, linear approximation theory (Jackson-type theorems) tells us that assuming a function $f^* \in L^2(\mu)$ is smooth, it may be decomposed onto the span of the $N$ first $(\eta_i)_{i\in\{1,\ldots,N\}}$ functions with decreasing coefficients $|b_i| \leq i^{-\lambda}$ for some $\lambda \geq 0$ that depends on the smoothness of $f^*$. For example the class of functions with bounded total variation may be decomposed with Fourier basis (in dimension 1) with coefficients $|b_i| \leq ||f||_V/(2\pi i)$. Thus here $\lambda = 1$. Other classes (such as Sobolev spaces) lead to larger values of $\lambda$ related to the order of differentiability.

By choosing $c_i = i^{-\lambda/2}$, we have $||\alpha^+||\sqrt{\mathbb{E}||\varphi||^2} \leq \sqrt{C}\sum_{i=1}^{N} i^{-\lambda}$. Thus if $\lambda > 1$, then this term is bounded by a constant that does not depend on $N$. If $\lambda = 1$ then it is bounded by $O(\log N)$, and if $0 < \lambda < 1$, then it is bounded by $O(N^{1-\lambda})$.

However *any* orthonormal basis, even rescaled, would not necessarily yield a small $||\alpha^+||\sqrt{\mathbb{E}||\varphi||^2}$ term (this is all the more true when the dimension of $\mathcal{X}$ is large). The desired property that the coefficients $(\alpha^+)_i$ of the decomposition of $f^*$ rapidly decrease to 0 indicates that hierarchical bases, such as wavelets, that would decompose the function at different scales, may be interesting.

**Wavelets:** Consider an infinite family of wavelets in $[0,1]$: $(\varphi_n^0) = (\varphi_{h,l}^0)$ (indexed by $n \geq 1$ or equivalently by the scale $h \geq 0$ and translation $0 \leq l \leq 2^h - 1$) where $\varphi_{h,l}^0(x) = 2^{h/2}\varphi_0(2^hx - l)$ and $\varphi_0$ is the mother wavelet. Then consider $N = 2^H$ features $(\varphi_{h,l})_{1\leq h\leq H}$ defined as the rescaled wavelets $\varphi_{h,l} \stackrel{\text{def}}{=} c_h 2^{-h/2}\varphi_{h,l}^0$, where $c_h > 0$ are some coefficients. Assume the mother wavelet is $\mathcal{C}^p$ (for $p \geq 1$), has at least $p$ vanishing moments, and that for all $h \geq 0$, $\sup_x \sum_l \varphi_0(2^hx - l)^2 \leq 1$. Then the following result (proof in the supplementary material) provides a bound on $\sup_{x\in\mathcal{X}}||\varphi(x)||^2$ (thus on $\sqrt{\mathbb{E}||\varphi(X)||^2}$) by a constant independent of $N$:

**Proposition 2** *Assume that $f^*$ is $(L,\gamma)$-Lipschitz (i.e. for all $v \in \mathcal{X}$ there exists a polynomial $p_v$ of degree $\lfloor\gamma\rfloor$ such that for all $u \in \mathcal{X}$, $|f(u) - p_v(u)| \leq L|u - v|^\gamma$) with $1/2 < \gamma \leq p$. Then setting $c_h = 2^{h(1-2\gamma)/4}$, we have $||\alpha^+||\sup_x||\varphi(x)|| \leq L\frac{2^\gamma}{1-2^{1/2-\gamma}}\int_0^1 |\varphi_0|$, which is independent of $N$.*

Notice that the Haar walevets has $p = 1$ vanishing moment but is not $\mathcal{C}^1$, thus the Proposition does not apply directly. However direct computations show that if $f^*$ is $L$-Lipschitz (i.e. $\gamma = 1$) then $\alpha_{h,l}^0 \leq L2^{-3h/2-2}$, and thus $||\alpha^+||\sup_x||\varphi(x)|| \leq \frac{L}{4(1-2^{-1/2})}$ with $c_h = 2^{-h/4}$.

### 4.2 Comparison with other methods

In the case when the factor $||\alpha^+||\sqrt{\mathbb{E}||\varphi(X)||^2}$ does not depend on $N$ (such as in the previous example), the bound (8) on the excess risk of CLSR states that the estimation error (assessed in terms of $\mathcal{F}_N$) of CLSR is $O(\log K/\sqrt{K})$. It is clear that whenever $N > \sqrt{K}$ (which is the case of interest here), this is better than the ordinary LSR in the initial domain, whose estimation error is $O(N \log K/K)$.

It is difficult to compare this result with LASSO (or the Dantzig selector that has similar properties [5]) for which an important aspect is to design sparse regression functions or to recover a solution assumed to be sparse. From [12, 15, 24] one deduces that under some assumptions, the estimation error of LASSO is of order $S\frac{\log N}{K}$ where $S$ is the sparsity (number of non-zero coefficients) of the best regressor $f^+$ in $\mathcal{F}_N$. If $S < \sqrt{K}$ then LASSO is more interesting than CLSR in terms of excess risk. Otherwise CLSR may be an interesting alternative although this method does not make any assumption about the sparsity of $f^+$ and its goal is not to recover a possible sparse $f^+$ but only to make good predictions. However, in some sense our method finds a sparse solution in the fact that the regression function $\widehat{g}_L$ lies in a space $\mathcal{G}_M$ of small dimension $M \ll N$ and can thus be expressed using only $M$ coefficients.

Now in terms of numerical complexity, CLSR requires $O(NK^{3/2})$ operations to build the matrix and compute the regression function, whereas according to [18], the (heuristical) complexity of the LASSO algorithm is $O(NK^2)$ in the best cases (assuming that the number of steps required for convergence is $O(K)$, which is not proved theoretically). Thus CLSR seems to be a good and simple competitor to LASSO.

## 5 Conclusion

We considered the case when the number of features $N$ is larger than the number of data $K$. The result stated in Theorem 1 enables to analyze the excess risk of any linear regression algorithm (LS or its penalized versions) performed in the compressed domain $\mathcal{G}_M$ versus in the initial space $\mathcal{F}_N$. In the compressed domain the estimation error is reduced but an additional (controlled) approximation error (when compared to the best regressor in $\mathcal{F}_N$) comes into the picture. In the case of LS regression, when the term $||\alpha^+||\sqrt{\mathbb{E}||\varphi(X)||^2}$ has a mild dependency on $N$, then by choosing a random subspace of dimension $M = O(\sqrt{K})$, CLSR has an estimation error (assessed in terms of $\mathcal{F}_N$) bounded by $O(\log K/\sqrt{K})$ and has numerical complexity $O(NK^{3/2})$.

In short, CLSR provides an alternative to usual penalization techniques where one first selects a random subspace of lower dimension and then performs an empirical risk minimizer in this subspace. Further work needs to be done to provide additional settings (when the space $\mathcal{X}$ is of dimension $> 1$) for which the term $||\alpha^+||\sqrt{\mathbb{E}||\varphi(X)||^2}$ is small.

**Acknowledgements:** The authors wish to thank Laurent Jacques for numerous comments and Alessandro Lazaric and Mohammad Ghavamzadeh for exciting discussions. This work has been supported by French National Research Agency (ANR) through COSINUS program (project EXPLO-RA, ANR-08-COSI-004).

## Footnotes

[1]In the full rank case, $\Phi^\dagger = (\Phi^T \Phi)^{-1} \Phi^T$ when $K \geq N$ and $\Phi^\dagger = \Phi^T (\Phi \Phi^T)^{-1}$ when $K \leq N$

## References

[1] Dimitris Achlioptas. Database-friendly random projections: Johnson-Lindenstrauss with binary coins. *Journal of Computer and System Sciences*, 66(4):671–687, June 2003.

[2] Nir Ailon and Bernard Chazelle. Approximate nearest neighbors and the fast Johnson-Lindenstrauss transform. In *STOC '06: Proceedings of the thirty-eighth annual ACM symposium on Theory of computing*, pages 557–563, New York, NY, USA, 2006. ACM.

[3] Jean-Yves Audibert and Olivier Catoni. Risk bounds in linear regression through pac-bayesian truncation. Technical Report HAL : hal-00360268, 2009.

[4] David Bau III and Lloyd N. Trefethen. *Numerical linear algebra*. Philadelphia: Society for Industrial and Applied Mathematics, 1997.

[5] Peter J. Bickel, Ya'acov Ritov, and Alexandre B. Tsybakov. Simultaneous analysis of Lasso and Dantzig selector. *To appear in Annals of Statistics*, 2008.

[6] Avrim Blum. Random projection, margins, kernels, and feature-selection. *Subspace, Latent Structure and Feature Selection*, pages 52–68, 2006.

[7] Robert Calderbank, Sina Jafarpour, and Robert Schapire. Compressed learning: Universal sparse dimensionality reduction and learning in the measurement domain. *Technical Report*, 2009.

[8] Emmanuel Candes and Terence Tao. The Dantzig selector: Statistical estimation when p is much larger than n. *Annals of Statistics*, 35:2313, 2007.

[9] Emmanuel J. Candes and Justin K. Romberg. Signal recovery from random projections. volume 5674, pages 76–86. SPIE, 2005.

[10] S. S. Chen, D. L. Donoho, and M. A. Saunders. Atomic decomposition by basis pursuit. *SIAM Journal on Scientific Computing*, 20:33–61, 1998.

[11] Mark A. Davenport, Michael B. Wakin, and Richard G. Baraniuk. Detection and estimation with compressive measurements. Technical Report TREE 0610, Department of Electrical and Computer Engineering, Rice University, 2006.

[12] E. Greenshtein and Y. Ritov. Persistency in high dimensional linear predictor-selection and the virtue of over-parametrization. *Bernoulli*, 10:971–988, 2004.

[13] L. Györfi, M. Kohler, A. Krzyżak, and H. Walk. *A distribution-free theory of nonparametric regression*. Springer-Verlag, 2002.

[14] Sham M. Kakade, Karthik Sridharan, and Ambuj Tewari. On the complexity of linear prediction: Risk bounds, margin bounds, and regularization. In Daphne Koller, Dale Schuurmans, Yoshua Bengio, and Leon Bottou, editors, *Neural Information Processing Systems*, pages 793–800. MIT Press, 2008.

[15] Yuval Nardi and Alessandro Rinaldo. On the asymptotic properties of the group Lasso estimator for linear models. *Electron. J. Statist.*, 2:605–633, 2008.

[16] D. Pollard. *Convergence of Stochastic Processes*. Springer Verlag, New York, 1984.

[17] Ali Rahimi and Benjamin Recht. Random features for large-scale kernel machines. *Neural Information Processing Systems*, 2007.

[18] Saharon Rosset and Ji Zhu. Piecewise linear regularized solution paths. *Annals of Statistics*, 35:1012, 2007.

[19] Robert Tibshirani. Regression shrinkage and selection via the Lasso. *Journal of the Royal Statistical Society, Series B*, 58:267–288, 1994.

[20] A. N. Tikhonov. Solution of incorrectly formulated problems and the regularization method. *Soviet Math Dokl 4*, pages 1035–1038, 1963.

[21] Yaakov Tsaig and David L. Donoho. Compressed sensing. *IEEE Trans. Inform. Theory*, 52:1289–1306, 2006.

[22] Vladimir N. Vapnik. *The nature of statistical learning theory*. Springer-Verlag New York, Inc., New York, NY, USA, 1995.

[23] Tong Zhang. Covering number bounds of certain regularized linear function classes. *Journal of Machine Learning Research*, 2:527–550, 2002.

[24] Tong Zhang. Some sharp performance bounds for least squares regression with L1 regularization. *To appear in Annals of Statistics*, 2009.

[25] Shuheng Zhou, John D. Lafferty, and Larry A. Wasserman. Compressed regression. In John C. Platt, Daphne Koller, Yoram Singer, and Sam T. Roweis, editors, *Neural Information Processing Systems*. MIT Press, 2007.

